# Non-Asymptotic Analysis of Stochastic Approximation Algorithms for Machine Learning

**Francis Bach**
INRIA - Sierra Project-team
Ecole Normale Supérieure, Paris, France
`francis.bach@ens.fr`

**Eric Moulines**
LTCI
Telecom ParisTech, Paris, France
`eric.moulines@enst.fr`

## Abstract

We consider the minimization of a convex objective function defined on a Hilbert space, which is only available through unbiased estimates of its gradients. This problem includes standard machine learning algorithms such as kernel logistic regression and least-squares regression, and is commonly referred to as a stochastic approximation problem in the operations research community. We provide a non-asymptotic analysis of the convergence of two well-known algorithms, stochastic gradient descent (a.k.a. Robbins-Monro algorithm) as well as a simple modification where iterates are averaged (a.k.a. Polyak-Ruppert averaging). Our analysis suggests that a learning rate proportional to the inverse of the number of iterations, while leading to the optimal convergence rate in the strongly convex case, is not robust to the lack of strong convexity or the setting of the proportionality constant. This situation is remedied when using slower decays together with averaging, robustly leading to the optimal rate of convergence. We illustrate our theoretical results with simulations on synthetic and standard datasets.

## 1 Introduction

The minimization of an objective function which is only available through unbiased estimates of the function values or its gradients is a key methodological problem in many disciplines. Its analysis has been attacked mainly in three communities: stochastic approximation [1, 2, 3, 4, 5, 6], optimization [7, 8], and machine learning [9, 10, 11, 12, 13, 14, 15]. The main algorithms which have emerged are stochastic gradient descent (a.k.a. Robbins-Monro algorithm), as well as a simple modification where iterates are averaged (a.k.a. Polyak-Ruppert averaging).

Traditional results from stochastic approximation rely on strong convexity and asymptotic analysis, but have made clear that a learning rate proportional to the inverse of the number of iterations, while leading to the optimal convergence rate in the strongly convex case, is not robust to the wrong setting of the proportionality constant. On the other hand, using slower decays together with averaging robustly leads to optimal convergence behavior (both in terms of rates and constants) [4, 5].

The analysis from the convex optimization and machine learning literatures however has focused on differences between strongly convex and non-strongly convex objectives, with learning rates and roles of averaging being different in these two cases [11, 12, 13, 14, 15].

A key desirable behavior of an optimization method is to be adaptive to the hardness of the problem, and thus one would like a single algorithm to work in all situations, favorable ones such as strongly convex functions and unfavorable ones such as non-strongly convex functions. In this paper, we unify the two types of analysis and show that (1) a learning rate proportional to the inverse of the number of iterations is not suitable because it is not robust to the setting of the proportionality constant and the lack of strong convexity, (2) the use of averaging with slower decays allows (close to) optimal rates in *all* situations.

More precisely, we make the following contributions:

- We provide a direct non-asymptotic analysis of stochastic gradient descent in a machine learning context (observations of real random functions defined on a Hilbert space) that includes

kernel least-squares regression and logistic regression (see Section 2), with strong convexity assumptions (Section 3) and without (Section 4).

 – We provide a non-asymptotic analysis of Polyak-Ruppert averaging [4, 5], with and without strong convexity (Sections 3.3 and 4.2). In particular, we show that slower decays of the learning rate, *together with averaging*, are crucial to *robustly* obtain fast convergence rates.

 – We illustrate our theoretical results through experiments on synthetic and non-synthetic examples in Section 5.

**Notation.** We consider a Hilbert space $\mathcal{H}$ with a scalar product $\langle \cdot, \cdot \rangle$. We denote by $\| \cdot \|$ the associated norm and use the same notation for the operator norm on bounded linear operators from $\mathcal{H}$ to $\mathcal{H}$, defined as $\|A\| = \sup_{\|x\| \leqslant 1} \|Ax\|$ (if $\mathcal{H}$ is a Euclidean space, then $\|A\|$ is the largest singular value of $A$). We also use the notation "w.p.1" to mean "with probability one". We denote by $\mathbb{E}$ the expectation or conditional expectation with respect to the underlying probability space.

## 2 Problem set-up

We consider a sequence of *convex differentiable random* functions $(f_n)_{n \geqslant 1}$ from $\mathcal{H}$ to $\mathbb{R}$. We consider the following recursion, starting from $\theta_0 \in \mathcal{H}$:

$$\forall n \geqslant 1, \ \ \theta_n = \theta_{n-1} - \gamma_n f_n'(\theta_{n-1}), \tag{1}$$

where $(\gamma_n)_{n \geqslant 1}$ is a deterministic sequence of positive scalars, which we refer to as the *learning rate sequence*. The function $f_n$ is assumed to be differentiable (see, e.g., [16] for definitions and properties of differentiability for functions defined on Hilbert spaces), and its gradient is an unbiased estimate of the gradient of a certain function $f$ we wish to minimize:

**(H1)** Let $(\mathcal{F}_n)_{n \geqslant 0}$ be an increasing family of $\sigma$-fields. $\theta_0$ is $\mathcal{F}_0$-measurable, and for each $\theta \in \mathcal{H}$, the random variable $f_n'(\theta)$ is square-integrable, $\mathcal{F}_n$-measurable and

$$\forall \theta \in \mathcal{H}, \ \ \forall n \geqslant 1, \ \ \mathbb{E}(f_n'(\theta)|\mathcal{F}_{n-1}) = f'(\theta), \ \text{w.p.1}. \tag{2}$$

For an introduction to martingales, $\sigma$-fields, and conditional expectations, see, e.g., [17]. Note that depending whether $\mathcal{F}_0$ is a trivial $\sigma$-field or not, $\theta_0$ may be random or not. Moreover, we could restrict Eq. (2) to be satisfied only for $\theta_{n-1}$ and $\theta^*$ (which is a global minimizer of $f$).

Given only the noisy gradients $f_n'(\theta_{n-1})$, the goal of stochastic approximation is to minimize the function $f$ with respect to $\theta$. Our assumptions include two usual situations, but also include many others (e.g., potentially, active learning):

 – **Stochastic approximation**: in the so-called Robbins-Monro setting, for all $\theta \in \mathcal{H}$ and $n \geqslant 1$, $f_n(\theta)$ may be expressed as $f_n(\theta) = f(\theta) + \langle \varepsilon_n, \theta \rangle$, where $(\varepsilon_n)_{n \geqslant 1}$ is a square-integrable martingale difference (i.e., such that $\mathbb{E}(\varepsilon_n|\mathcal{F}_{n-1}) = 0$), which corresponds to a noisy observation $f'(\theta_{n-1}) + \varepsilon_n$ of the gradient $f'(\theta_{n-1})$.

 – **Learning from i.i.d. observations**: for all $\theta \in \mathcal{H}$ and $n \geqslant 1$, $f_n(\theta) = \ell(\theta, z_n)$ where $z_n$ is an i.i.d. sequence of observations in a measurable space $\mathcal{Z}$ and $\ell : \mathcal{H} \times \mathcal{Z}$ is a loss function. Then $f(\theta)$ is the generalization error of the predictor defined by $\theta$. Classical examples are least-squares or logistic regression (linear or non-linear through kernel methods [18, 19]), where $f_n(\theta) = \frac{1}{2}(\langle x_n, \theta \rangle - y_n)^2$, or $f_n(\theta) = \log[1 + \exp(-y_n \langle x_n, \theta \rangle)]$, for $x_n \in \mathcal{H}$, and $y_n \in \mathbb{R}$ (or $\{-1, 1\}$ for logistic regression).

Throughout this paper, unless otherwise stated, we assume that each function $f_n$ is convex and *smooth*, following the traditional definition of smoothness from the optimization literature, i.e., Lipschitz-continuity of the gradients (see, e.g., [20]). However, we make two slightly different assumptions: **(H2)** where the function $\theta \mapsto \mathbb{E}(f_n'(\theta)|\mathcal{F}_{n-1})$ is Lipschitz-continuous in quadratic mean and a strengthening of this assumption, **(H2')** in which $\theta \mapsto f_n'(\theta)$ is almost surely Lipschitz-continuous.

**(H2)** For each $n \geqslant 1$, the function $f_n$ is almost surely convex, differentiable, and:

$$\forall n \geqslant 1, \ \forall \theta_1, \theta_2 \in \mathcal{H}, \ \ \mathbb{E}(\|f_n'(\theta_1) - f_n'(\theta_2)\|^2|\mathcal{F}_{n-1}) \leqslant L^2\|\theta_1 - \theta_2\|^2, \quad \text{w.p.1}. \tag{3}$$

**(H2')** For each $n \geqslant 1$, the function $f_n$ is almost surely convex, differentiable with Lipschitz-continuous gradient $f_n'$, with constant $L$, that is:

$$\forall n \geqslant 1, \ \forall \theta_1, \theta_2 \in \mathcal{H}, \ \ \|f_n'(\theta_1) - f_n'(\theta_2)\| \leqslant L\|\theta_1 - \theta_2\|, \quad \text{w.p.1}. \tag{4}$$

If $f_n$ is twice differentiable, this corresponds to having the operator norm of the Hessian operator of $f_n$ bounded by $L$. For least-squares or logistic regression, if we assume that $(\mathbb{E}\|x_n\|^4)^{1/4} \leqslant R$ for all $n \in \mathbb{N}$, then we may take $L = R^2$ (or even $L = R^2/4$ for logistic regression) for assumption (**H2**), while for assumption (**H2'**), we need to have an almost sure bound $\|x_n\| \leqslant R$.

## 3 Strongly convex objectives

In this section, following [21], we make the additional assumption of strong convexity of $f$, but not of all functions $f_n$ (see [20] for definitions and properties of such functions):

(**H3**) The function $f$ is strongly convex with respect to the norm $\|\cdot\|$, with convexity constant $\mu > 0$. That is, for all $\theta_1, \theta_2 \in \mathcal{H}$, $f(\theta_1) \geqslant f(\theta_2) + \langle f'(\theta_2), \theta_1 - \theta_2 \rangle + \frac{\mu}{2}\|\theta_1 - \theta_2\|^2$.

Note that (**H3**) simply needs to be satisfied for $\theta_2 = \theta^*$ being the unique global minimizer of $f$ (such that $f'(\theta^*) = 0$). In the context of machine learning (least-squares or logistic regression), assumption (**H3**) is satisfied as soon as $\frac{\mu}{2}\|\theta\|^2$ is used as an additional regularizer. For all strongly convex losses (e.g., least-squares), it is also satisfied as soon as the expectation $\mathbb{E}(x_n \otimes x_n)$ is invertible. Note that this implies that the problem is finite-dimensional, otherwise, the expectation is a compact covariance operator, and hence non-invertible (see, e.g., [22] for an introduction to covariance operators). For non-strongly convex losses such as the logistic loss, $f$ can never be strongly convex unless we restrict the domain of $\theta$ (which we do in Section 3.2). Alternatively to restricting the domain, replacing the logistic loss $u \mapsto \log(1 + e^{-u})$ by $u \mapsto \log(1 + e^{-u}) + \varepsilon u^2/2$, for some small $\varepsilon > 0$, makes it strongly convex in low-dimensional settings.

By strong convexity of $f$, if we assume (**H3**), then $f$ attains its global minimum at a unique vector $\theta^* \in \mathcal{H}$ such that $f'(\theta^*) = 0$. Moreover, we make the following assumption (in the context of stochastic approximation, it corresponds to $\mathbb{E}(\|\varepsilon_n\|^2|\mathcal{F}_{n-1}) \leqslant \sigma^2$):

(**H4**) There exists $\sigma^2 \in \mathbb{R}_+$ such that for all $n \geqslant 1$, $\mathbb{E}(\|f_n'(\theta^*)\|^2|\mathcal{F}_{n-1}) \leqslant \sigma^2$, w.p.1.

### 3.1 Stochastic gradient descent

Before stating our first theorem (see proof in [23]), we introduce the following family of functions $\varphi_\beta : \mathbb{R}_+ \setminus \{0\} \to \mathbb{R}$ given by:
$$\varphi_\beta(t) = \begin{cases} \frac{t^\beta - 1}{\beta} & \text{if } \beta \neq 0, \\ \log t & \text{if } \beta = 0. \end{cases}$$
The function $\beta \mapsto \varphi_\beta(t)$ is continuous for all $t > 0$. Moreover, for $\beta > 0$, $\varphi_\beta(t) < \frac{t^\beta}{\beta}$, while for $\beta < 0$, we have $\varphi_\beta(t) < \frac{1}{-\beta}$ (both with asymptotic equality when $t$ is large).

**Theorem 1 (Stochastic gradient descent, strong convexity)** *Assume* (**H1,H2,H3,H4**). *Denote* $\delta_n = \mathbb{E}\|\theta_n - \theta^*\|^2$, *where* $\theta_n \in \mathcal{H}$ *is the $n$-th iterate of the recursion in Eq. (1), with* $\gamma_n = Cn^{-\alpha}$. *We have, for* $\alpha \in [0, 1]$:

$$\delta_n \leqslant \begin{cases} 2\exp\left(4L^2C^2\varphi_{1-2\alpha}(n)\right)\exp\left(-\frac{\mu C}{4}n^{1-\alpha}\right)\left(\delta_0 + \frac{\sigma^2}{L^2}\right) + \frac{4C\sigma^2}{\mu n^\alpha}, & \text{if } 0 \leqslant \alpha < 1, \\ \frac{\exp(2L^2C^2)}{n^{\mu C}}\left(\delta_0 + \frac{\sigma^2}{L^2}\right) + 2\sigma^2 C^2 \frac{\varphi_{\mu C/2 - 1}(n)}{n^{\mu C/2}}, & \text{if } \alpha = 1. \end{cases} \quad (5)$$

**Sketch of proof.** Under our assumptions, it can be shown that $(\delta_n)$ satisfies the following recursion:
$$\delta_n \leqslant (1 - 2\mu\gamma_n + 2L^2\gamma_n^2)\delta_{n-1} + 2\sigma^2\gamma_n^2. \quad (6)$$
Note that it also appears in [3, Eq. (2)] under different assumptions. Using this *deterministic* recursion, we then derive bounds using classical techniques from stochastic approximation [2], but in a non-asymptotic way, by deriving explicit upper-bounds.

**Related work.** To the best of our knowledge, this non-asymptotic bound, which depends explicitly upon the parameters of the problem, is novel (see [1, Theorem 1, Electronic companion paper] for a simpler bound with no such explicit dependence). It shows in particular that there is convergence in quadratic mean for any $\alpha \in (0, 1]$. Previous results from the stochastic approximation literature have focused mainly on almost sure convergence of the sequence of iterates. Almost-sure convergence requires that $\alpha > 1/2$, with counter-examples for $\alpha < 1/2$ (see, e.g., [2] and references therein).

**Bound on function values.** The bounds above imply a corresponding a bound on the functions values. Indeed, under assumption **(H2)**, it may be shown that $\mathbb{E}[f(\theta_n) - f(\theta^*)] \leqslant \frac{L}{2}\delta_n$ (see proof in [23]).

**Tightness for quadratic functions.** Since the deterministic recursion in Eq. (6) is an equality for quadratic functions $f_n$, the result in Eq. (5) is optimal (up to constants). Moreover, our results are consistent with the asymptotic results from [6].

**Forgetting initial conditions.** Bounds depend on the initial condition $\delta_0 = \mathbb{E}\left[\|\theta_0 - \theta^*\|^2\right]$ and the variance $\sigma^2$ of the noise term. The initial condition is forgotten sub-exponentially fast for $\alpha \in (0, 1)$, but not for $\alpha = 1$. For $\alpha < 1$, the asymptotic term in the bound is $\frac{4C\sigma^2}{\mu n^\alpha}$.

**Behavior for $\alpha = 1$.** For $\alpha = 1$, we have $\frac{\varphi_{\mu C/2-1}(n)}{n^{\mu C/2}} \leqslant \frac{1}{\mu C/2-1}\frac{1}{n}$ if $C\mu > 2$, $\frac{\varphi_{\mu C/2-1}(n)}{n^{\mu C/2}} = \frac{\log n}{n}$ if $C\mu = 2$ and $\frac{\varphi_{\mu C/2-1}(n)}{n^{\mu C/2}} \leqslant \frac{1}{1-\mu C/2}\frac{1}{n^{\mu C/2}}$ if $C\mu > 2$. Therefore, for $\alpha = 1$, the choice of $C$ is critical, as already noticed by [8]: too small $C$ leads to convergence at arbitrarily small rate of the form $n^{-\mu C/2}$, while too large $C$ leads to explosion due to the initial condition. This behavior is confirmed in simulations in Section 5.

**Setting $C$ too large.** There is a potentially catastrophic term when $C$ is chosen too large, i.e., $\exp\left(4L^2C^2\varphi_{1-2\alpha}(n)\right)$, which leads to an increasing bound when $n$ is small. Note that for $\alpha < 1$, this catastrophic term is in front of a sub-exponentially decaying factor, so its effect is mitigated once the term in $n^{1-\alpha}$ takes over $\varphi_{1-2\alpha}(n)$, and the transient term stops increasing. Moreover, the asymptotic term is not involved in it (which is also observed in simulations in Section 5).

**Minimax rate.** Note finally, that the asymptotic convergence rate in $O(n^{-1})$ matches optimal asymptotic minimax rate for stochastic approximation [24, 25]. Note that there is no explicit dependence on dimension; this dependence is implicit in the definition of the constants $\mu$ and $L$.

## 3.2 Bounded gradients

In some cases such as logistic regression, we also have a uniform upper-bound on the gradients, i.e., we assume (note that in Theorem 2, this assumption replaces both **(H2)** *and* **(H4)**).

**(H5)** For each $n \geqslant 1$, almost surely, the function $f_n$ if convex, differentiable and has gradients uniformly bounded by $B$ on the ball of center 0 and radius $D$, i.e., for all $\theta \in \mathcal{H}$ and all $n > 0$, $\|\theta\| \leqslant D \Rightarrow \|f'_n(\theta)\| \leqslant B$.

Note that no function may be strongly convex and Lipschitz-continuous (i.e., with uniformly bounded gradients) over the entire Hilbert space $\mathcal{H}$. Moreover, if **(H2')** is satisfied, then we may take $D = \|\theta^*\|$ and $B = LD$. The next theorem shows that with a slight modification of the recursion in Eq. (1), we get simpler bounds than the ones obtained in Theorem 1, obtaining a result which already appeared in a simplified form [8] (see proof in [23]):

**Theorem 2 (Stochastic gradient descent, strong convexity, bounded gradients)** *Assume* **(H1,H3,H5)**. *Denote* $\delta_n = \mathbb{E}\left[\|\theta_n - \theta^*\|^2\right]$, *where* $\theta_n \in \mathcal{H}$ *is the $n$-th iterate of the following recursion:*

$$\forall n \geqslant 1, \;\; \theta_n = \Pi_D[\theta_{n-1} - \gamma_n f'_n(\theta_{n-1})], \tag{7}$$

*where $\Pi_D$ is the orthogonal projection operator on the ball $\{\theta : \|\theta\| \leqslant D\}$. Assume $\|\theta^*\| \leqslant D$. If $\gamma_n = Cn^{-\alpha}$, we have, for $\alpha \in [0, 1]$:*

$$\delta_n \leqslant \begin{cases} \left(\delta_0 + B^2C^2\varphi_{1-2\alpha}(n)\right)\exp\left(-\frac{\mu C}{2}n^{1-\alpha}\right) + \frac{2B^2C^2}{\mu n^\alpha}, & \text{if } \alpha \in [0, 1); \\ \delta_0 n^{-\mu C} + 2B^2C^2 n^{-\mu C}\varphi_{\mu C-1}(n), & \text{if } \alpha = 1. \end{cases} \tag{8}$$

The proof follows the same lines than for Theorem 1, but with the deterministic recursion $\delta_n \leqslant (1 - 2\mu\gamma_n)\delta_{n-1} + B^2\gamma_n^2$. Note that we obtain the same asymptotic terms than for Theorem 1 (but $B$ replaces $\sigma$). Moreover, the bound is simpler (no explosive multiplicative factors), but it requires to know $D$ in advance, while Theorem 1 does not. Note that because we have only assumed Lipschitz-continuity, we obtain a bound on function values of order $O(n^{-\alpha/2})$, which is sub-optimal. For bounds directly on function values, see [26].

### 3.3 Polyak-Ruppert averaging

We now consider $\bar{\theta}_n = \frac{1}{n}\sum_{k=0}^{n-1}\theta_k$ and, following [4, 5], we make extra assumptions regarding the smoothness of each $f_n$ and the fourth-order moment of the driving noise:

(**H6**) For each $n \geqslant 1$, the function $f_n$ is almost surely twice differentiable with Lipschitz-continuous Hessian operator $f_n''$, with Lipschitz constant $M$. That is, for all $\theta_1, \theta_2 \in \mathcal{H}$ and for all $n \geqslant 1$, $\|f_n''(\theta_1) - f_n''(\theta_2)\| \leqslant M\|\theta_1 - \theta_2\|$, where $\|\cdot\|$ is the operator norm.

Note that (**H6**) needs only to be satisfied for $\theta_2 = \theta^*$. For least-square regression, we have $M = 0$, while for logistic regression, we have $M = R^3/4$.

(**H7**) There exists $\tau \in \mathbb{R}_+$, such that for each $n \geqslant 1$, $\mathbb{E}(\|f_n'(\theta^*)\|^4|\mathcal{F}_{n-1}) \leqslant \tau^4$ almost surely. Moreover, there exists a nonnegative self-adjoint operator $\Sigma$ such that for all $n$, $\mathbb{E}(f_n'(\theta^*) \otimes f_n'(\theta^*)|\mathcal{F}_{n-1}) \preccurlyeq \Sigma$ almost-surely.

The operator $\Sigma$ (which always exists as soon as $\tau$ is finite) is here to characterize precisely the variance term, which will be independent of the learning rate sequence $(\gamma_n)$, as we now show:

**Theorem 3 (Averaging, strong convexity)** *Assume* (**H1, H2', H3, H4, H6, H7**)*. Then, for $\bar{\theta}_n = \frac{1}{n}\sum_{k=0}^{n-1}\theta_k$ and $\alpha \in (0,1)$, we have:*

$$
\left(\mathbb{E}\|\bar{\theta}_n - \theta^*\|^2\right)^{1/2} \leqslant \frac{\left[\operatorname{tr} f''(\theta^*)^{-1}\Sigma f''(\theta^*)^{-1}\right]^{1/2}}{\sqrt{n}} + \frac{6\sigma}{\mu C^{1/2}}\frac{1}{n^{1-\alpha/2}} + \frac{MC\tau^2}{2\mu^{3/2}}(1+(\mu C)^{1/2})\frac{\varphi_{1-\alpha}(n)}{n}
$$

$$
+ \frac{4LC^{1/2}}{\mu}\frac{\varphi_{1-\alpha}(n)^{1/2}}{n} + \frac{8A}{n\mu^{1/2}}\left(\frac{1}{C}+L\right)\left(\delta_0 + \frac{\sigma^2}{L^2}\right)^{1/2}
$$

$$
+ \frac{5MC^{1/2}\tau}{2n\mu}A\exp\left(24L^4C^4\right)\left(\delta_0 + \frac{\mu\mathbb{E}\left[\|\theta_0 - \theta^*\|^4\right]}{20C\tau^2} + 2\tau^2C^3\mu + 8\tau^2C^2\right)^{1/2}, \tag{9}
$$

*where $A$ is a constant that depends only on $\mu$, $C$, $L$ and $\alpha$.*

**Sketch of proof.** Following [4], we start from Eq. (1), write it as $f_n'(\theta_{n-1}) = \frac{1}{\gamma_n}(\theta_{n-1} - \theta_n)$, and notice that (a) $f_n'(\theta_{n-1}) \approx f_n'(\theta^*) + f''(\theta^*)(\theta_{n-1} - \theta^*)$, (b) $f_n'(\theta^*)$ has zero mean and behaves like an i.i.d. sequence, and (c) $\frac{1}{n}\sum_{k=1}^{n}\frac{1}{\gamma_k}(\theta_{k-1} - \theta_k)$ turns out to be negligible owing to a summation by parts and to the bound obtained in Theorem 1. This implies that $\bar{\theta}_n - \theta^*$ behaves like $-\frac{1}{n}\sum_{k=1}^{n}f''(\theta^*)^{-1}f_k'(\theta^*)$. Note that we obtain a bound on the *root* mean square error.

**Forgetting initial conditions.** There is no sub-exponential forgetting of initial conditions, but rather a decay at rate $O(n^{-2})$ (last two lines in Eq. (9)). This is a known problem which may slow down the convergence, a common practice being to start averaging after a certain number of iterations [2]. Moreover, the constant $A$ may be large when $LC$ is large, thus the catastrophic terms are more problematic than for stochastic gradient descent, because they do not appear in front of sub-exponentially decaying terms (see [23]). This suggests to take $CL$ small.

**Asymptotically leading term.** When $M > 0$ and $\alpha > 1/2$, the asymptotic term for $\delta_n$ is independent of $(\gamma_n)$ and of order $O(n^{-1})$. Thus, averaging allows to get from the slow rate $O(n^{-\alpha})$ to the optimal rate $O(n^{-1})$. The next two leading terms (in the first line) have order $O(n^{\alpha-2})$ and $O(n^{-2\alpha})$, suggesting the setting $\alpha = 2/3$ to make them equal. When $M = 0$ (quadratic functions), the leading term has rate $O(n^{-1})$ for all $\alpha \in (0,1)$ (with then a contribution of the first term in the second line).

**Case $\alpha = 1$.** We get a simpler bound by directly averaging the bound in Theorem 1, which leads to an unchanged rate of $n^{-1}$, i.e., averaging is not key for $\alpha = 1$, and does not solve the robustness problem related to the choice of $C$ or the lack of strong convexity.

**Leading term independent of $(\gamma_n)$.** The term in $O(n^{-1})$ does not depend on $\gamma_n$. Moreover, as noticed in the stochastic approximation literature [4], in the context of learning from i.i.d. observations, this is exactly the Cramer-Rao bound (see, e.g., [27]), and thus the leading term is asymptotically optimal. Note that no explicit Hessian inversion has been performed to achieve this bound.

**Relationship with prior work on online learning.** There is no clear way of adding a bounded gradient assumption in the general case $\alpha \in (0,1)$, because the proof relies on the recursion without projections, but for $\alpha = 1$, the rate of $O(n^{-1})$ (up to a logarithmic term) can be achieved in the more general framework of online learning, where averaging is key to deriving bounds for stochastic approximation from regret bounds. Moreover, bounds are obtained in high probability rather than simply in quadratic mean (see, e.g., [11, 12, 13, 14, 15]).

# 4 Non-strongly convex objectives

In this section, we do not assume that the function $f$ is strongly convex, but we replace (**H3**) by:

(**H8**) The function $f$ attains its global minimum at a certain $\theta^* \in \mathcal{H}$ (which may not be unique).

In the machine learning scenario, this essentially implies that the best predictor is in the function class we consider.[1] In the following theorem, since $\theta^*$ is not unique, we only derive a bound on function values. Not assuming strong convexity is essential in practice to make sure that algorithms are robust and *adaptive* to the hardness of the learning or optimization problem (much like gradient descent is).

## 4.1 Stochastic gradient descent

The following theorem is shown in a similar way to Theorem 1; we first derive a deterministic recursion, which we analyze with novel tools compared to the non-stochastic case (see details in [23]), obtaining new convergence rates for non-averaged stochastic gradient descent :

**Theorem 4 (Stochastic gradient descent, no strong convexity)** *Assume* (**H1,H2',H4,H8**). *Then, if $\gamma_n = Cn^{-\alpha}$, for $\alpha \in [1/2, 1]$, we have:*

$$\mathbb{E}\left[f(\theta_n) - f(\theta^*)\right] \leqslant \frac{1}{C}\left(\delta_0 + \frac{\sigma^2}{L^2}\right)\exp\left(4L^2C^2\varphi_{1-2\alpha}(n)\right)\frac{1 + 4L^{3/2}C^{3/2}}{\min\{\varphi_{1-\alpha}(n), \varphi_{\alpha/2}(n)\}}. \tag{10}$$

When $\alpha = 1/2$, the bound goes to zero only when $LC < 1/4$, at rates which can be arbitrarily slow. For $\alpha \in (1/2, 2/3)$, we get convergence at rate $O(n^{-\alpha/2})$, while for $\alpha \in (2/3, 1)$, we get a convergence rate of $O(n^{\alpha-1})$. For $\alpha = 1$, the upper bound is of order $O((\log n)^{-1})$, which may be very slow (but still convergent). The rate of convergence changes at $\alpha = 2/3$, where we get our best rate $O(n^{-1/3})$, which does not match the minimax rate of $O(n^{-1/2})$ for stochastic approximation in the non-strongly convex case [25]. These rates for stochastic gradient descent without strong convexity assumptions are new and we conjecture that they are asymptotically minimax optimal (for stochastic gradient descent, not for stochastic approximation). Nevertheless, the proof of this result falls out of the scope of this paper.

If we further assume that we have all gradients bounded by $B$ (that is, we assume $D = \infty$ in (**H5**)), then, we have the following theorem, which allows $\alpha \in (1/3, 1/2)$ with rate $O(n^{-3\alpha/2+1/2})$:

**Theorem 5 (Stochastic gradient descent, no strong convexity, bounded gradients)** *Assume* (**H1**, **H2'**, **H5**, **H8**). *Then, if $\gamma_n = Cn^{-\alpha}$, for $\alpha \in [1/3, 1]$, we have:*

$$\mathbb{E}\left[f(\theta_n) - f(\theta^*)\right] \leqslant \begin{cases} \left(\delta_0 + B^2C^2\varphi_{1-2\alpha}(n)\right)\frac{1 + 4L^{1/2}C^{1/2}}{C\min\{\varphi_{1-\alpha}(n), \varphi_{\alpha/2}(n)\}}, & \text{if } \alpha \in [1/2, 1], \\ \frac{2}{C}(\delta_0 + B^2C^2)^{1/2}\frac{(1 + 4L^{1/2}BC^{3/2})}{(1-2\alpha)^{1/2}\varphi_{3\alpha/2-1/2}(n)}, & \text{if } \alpha \in [1/3, 1/2]. \end{cases} \tag{11}$$

## 4.2 Polyak-Ruppert averaging

Averaging in the context of non-strongly convex functions has been studied before, in particular in the optimization and machine learning literature, and the following theorems are similar in spirit to earlier work [7, 8, 13, 14, 15]:

**Theorem 6 (averaging, no strong convexity)** *Assume* (**H1,H2',H4,H8**). *Then, if $\gamma_n = Cn^{-\alpha}$, for $\alpha \in [1/2, 1]$, we have*

$$\mathbb{E}\left[f(\bar{\theta}_n) - f(\theta^*)\right] \leqslant \frac{1}{C}\left(\delta_0 + \frac{\sigma^2}{L^2}\right)\frac{\exp\left(2L^2C^2\varphi_{1-2\alpha}(n)\right)}{n^{1-\alpha}}\left[1 + (2LC)^{1+\frac{1}{\alpha}}\right] + \frac{\sigma^2C}{2n}\varphi_{1-\alpha}(n). \tag{12}$$

If $\alpha = 1/2$, then we only have convergence under $LC < 1/4$ (as in Theorem 4), with potentially slow rate, while for $\alpha > 1/2$, we have a rate of $O(n^{-\alpha})$, with otherwise similar behavior than for the strongly convex case with no bounded gradients. Here, averaging has allowed the rate to go from $O(\max\{n^{\alpha-1}, n^{-\alpha/2}\})$ to $O(n^{-\alpha})$.

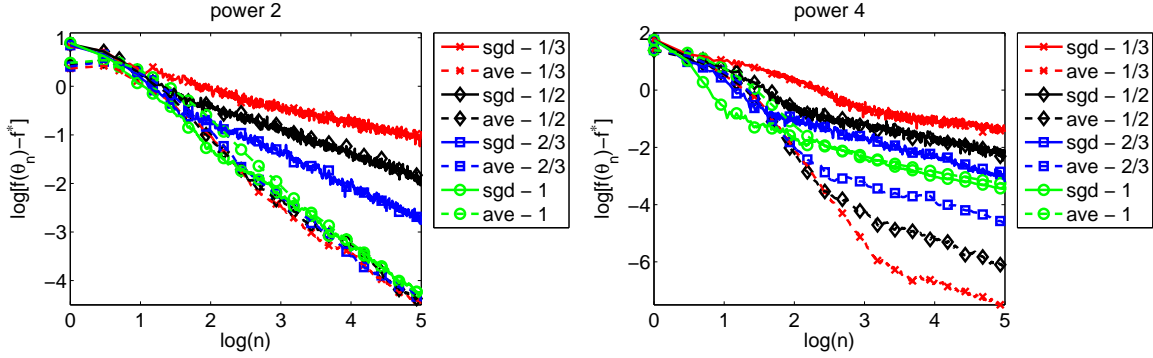

Figure 1: Robustness to lack of strong convexity for different learning rates and stochastic gradient (sgd) and Polyak-Ruppert averaging (ave). From left to right: $f(\theta) = |\theta|^2$ and $f(\theta) = |\theta|^4$, (between $-1$ and $1$, affine outside of $[-1, 1]$, continuously differentiable). See text for details.

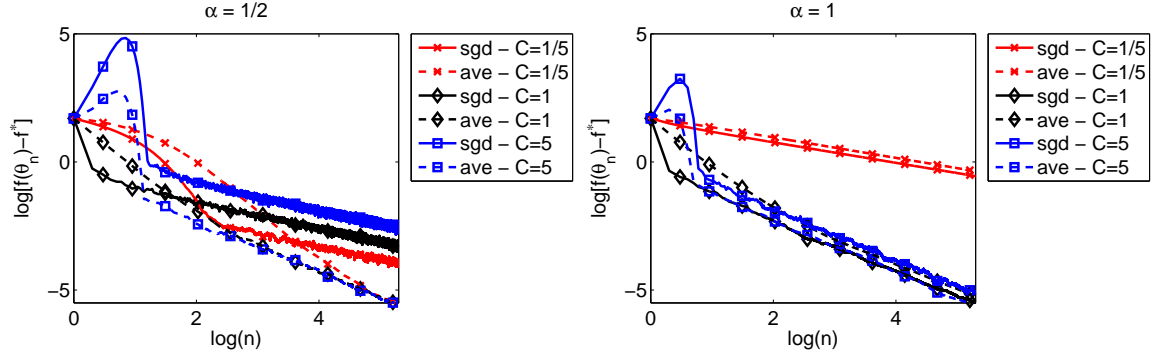

Figure 2: Robustness to wrong constants for $\gamma_n = Cn^{-\alpha}$. Left: $\alpha = 1/2$, right: $\alpha = 1$. See text for details. Best seen in color.

**Theorem 7 (averaging, no strong convexity, bounded gradients)** *Assume* (**H1,H5,H8**). *If* $\gamma_n = Cn^{-\alpha}$, *for* $\alpha \in [0, 1]$, *we have*

$$\mathbb{E}\left[f(\bar{\theta}_n) - f(\theta^*)\right] \leqslant \frac{n^{\alpha-1}}{2C}(\delta_0 + C^2 B^2 \varphi_{1-2\alpha}(n)) + \frac{B^2}{2n}\varphi_{1-\alpha}(n). \tag{13}$$

With the bounded gradient assumption (and in fact without smoothness), we obtain the minimax asymptotic rate $O(n^{-1/2})$ up to logarithmic terms [25] for $\alpha = 1/2$, and, for $\alpha < 1/2$, the rate $O(n^{-\alpha})$ while for $\alpha > 1/2$, we get $O(n^{\alpha-1})$. Here, averaging has also allowed to increase the range of $\alpha$ which ensures convergence, to $\alpha \in (0, 1)$.

## 5 Experiments

**Robustness to lack of strong convexity.** Define $f : \mathbb{R} \to \mathbb{R}$ as $|\theta|^q$ for $|\theta| \leqslant 1$ and extended into a continuously differentiable function, affine outside of $[-1, 1]$. For all $q > 1$, we have a convex function with Lipschitz-continuous gradient with constant $L = q(q-1)$. It is strongly convex around the origin for $q \in (1, 2]$, but its second derivative vanishes for $q > 2$. In Figure 1, we plot in log-log scale the average of $f(\theta_n) - f(\theta^*)$ over 100 replications of the stochastic approximation problem (with i.i.d. Gaussian noise of standard deviation 4 added to the gradient). For $q = 2$ (left plot), where we locally have a strongly convex case, all learning rates lead to good estimation with decay proportional to $\alpha$ (as shown in Theorem 1), while for the averaging case, all reach the exact same convergence rate (as shown in Theorem 3). However, for $q = 4$ where strong convexity does not hold (right plot), without averaging, $\alpha = 1$ is still fastest but becomes the slowest after averaging; on the contrary, illustrating Section 4, slower decays (such as $\alpha = 1/2$) leads to faster convergence when averaging is used. Note also the reduction in variability for the averaged iterations.

**Robustness to wrong constants.** We consider the function on the real line $f$, defined as $f(\theta) = \frac{1}{2}|\theta|^2$ and consider standard i.i.d. Gaussian noise on the gradients. In Figure 2, we plot the average performance over 100 replications, for various values of $C$ and $\alpha$. Note that for $\alpha = 1/2$ (left plot), the 3 curves for stochastic gradient descent end up being aligned and equally spaced, corroborating a rate proportional to $C$ (see Theorem 1). Moreover, when averaging for $\alpha = 1/2$, the error ends up

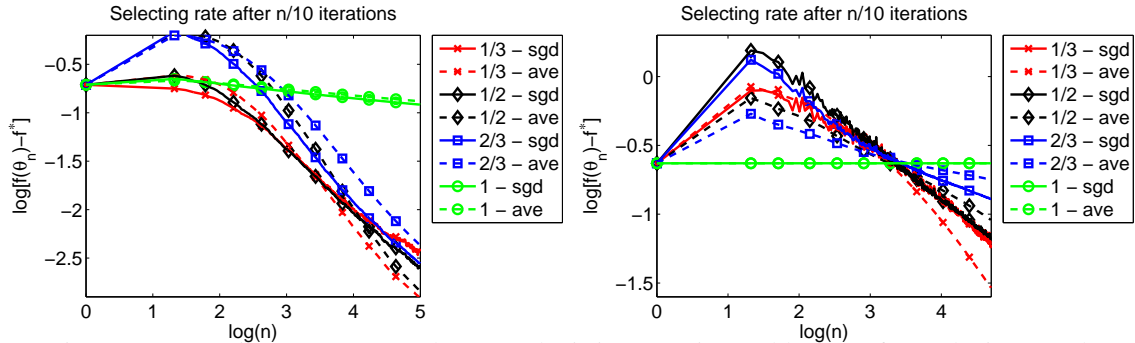

Figure 3: Comparison on non strongly convex logistic regression problems. Left: synthetic example, right: "alpha" dataset. See text for details. Best seen in color.

being independent of $C$ and $\alpha$ (see Theorem 3). Finally, when $C$ is too large, there is an explosion (up to $10^5$), hinting at the potential instability of having $C$ too large. For $\alpha = 1$ (right plot), if $C$ is too small, convergence is very slow (and not at the rate $n^{-1}$), as already observed (see, e.g., [8, 6]).

**Medium-scale experiments with linear logistic regression.** We consider two situations where $\mathcal{H} = \mathbb{R}^p$: (a) the "alpha" dataset from the Pascal large scale learning challenge (http://largescale.ml.tu-berlin.de/), for which $p = 500$ and $n = 50000$, and (b) a synthetic example where $p = 100$, $n = 100000$; we generate the input data i.i.d. from a multivariate Gaussian distribution with mean zero and a covariance matrix sampled from a Wishart distribution with $p$ degrees of freedom (thus with potentially bad condition number), and the output is obtained through a classification by a random hyperplane. For different values of $\alpha$, we choose $C$ in an adaptive way where we consider the lowest test error after $n/10$ iterations, and report results in Figure 3. In experiments reported in [23], we also consider $C$ equal to $1/L$ suggested by our analysis to avoid large constants, for which the convergence speed is very slow, suggesting that our global bounds involving the Lipschitz constants may be locally far too pessimistic and that designing a truly adaptive sequence $(\gamma_n)$ instead of a fixed one is a fruitful avenue for future research.

## 6 Conclusion

In this paper, we have provided a non-asymptotic analysis of stochastic gradient, as well as its averaged version, for various learning rate sequences of the form $\gamma_n = Cn^{-\alpha}$ (see summary of results in Table 1). Following earlier work from the optimization, machine learning and stochastic approximation literatures, our analysis highlights that $\alpha = 1$ is not robust to the choice of $C$ and to the actual difficulty of the problem (strongly convex or not). However, when using averaging with $\alpha \in (1/2, 1)$, we get, both in strongly convex and non-strongly convex situation, close to optimal rates of convergence. Moreover, we highlight the fact that problems with bounded gradients have better behaviors, i.e., logistic regression is easier to optimize than least-squares regression.

Our work can be extended in several ways: first, we have focused on results in quadratic mean and we expect that some of our results can be extended to results in high probability (in the line of [13, 3]). Second, we have focused on differentiable objectives, but the extension to objective functions with a differentiable stochastic part and a non-differentiable deterministic (in the line of [14]) would allow an extension to sparse methods.

**Acknowledgements.** Francis Bach was partially supported by the European Research Council (SIERRA Project). We thank Mark Schmidt and Nicolas Le Roux for helpful discussions.

| $\alpha$ | SGD $\mu, L$ | SGD $\mu, B$ | SGD $L$ | SGD $L, B$ | Aver. $\mu, L$ | Aver. $L$ | Aver. $B$ |
|---|---|---|---|---|---|---|---|
| (0 , 1/3) | $\alpha$ | $\alpha$ | $\times$ | $\times$ | $2\alpha$ | $\times$ | $\alpha$ |
| (1/3 , 1/2) | $\alpha$ | $\alpha$ | $\times$ | $(3\alpha - 1)/2$ | $2\alpha$ | $\times$ | $\alpha$ |
| (1/2 , 2/3) | $\alpha$ | $\alpha$ | $\alpha/2$ | $\alpha/2$ | $1$ | $1 - \alpha$ | $1 - \alpha$ |
| (2/3 , 1) | $\alpha$ | $\alpha$ | $1 - \alpha$ | $1 - \alpha$ | $1$ | $1 - \alpha$ | $1 - \alpha$ |

Table 1: Summary of results: For stochastic gradient descent (SGD) or Polyak-Ruppert averaging (Aver.), we provide their rates of convergence of the form $n^{-\beta}$ corresponding to learning rate sequences $\gamma_n = Cn^{-\alpha}$, where $\beta$ is shown as a function of $\alpha$. For each method, we list the main assumptions ($\mu$: strong convexity, $L$: bounded Hessian, $B$: bounded gradients).

## Footnotes

[1] For least-squares regression with kernels, where $f_n(\theta) = \frac{1}{2}(y_n - \langle\theta, \Phi(x_n)\rangle)^2$, with $\Phi(x_n)$ being the feature map associated with a reproducing kernel Hilbert space $\mathcal{H}$ with universal kernel [28], then we need that $x \mapsto \mathbb{E}(Y|X=x)$ is a function within the RKHS. Taking care of situations where this is not true is clearly of importance but out of the scope of this paper.

# References

[1] M. N. Broadie, D. M. Cicek, and A. Zeevi. General bounds and finite-time improvement for stochastic approximation algorithms. Technical report, Columbia University, 2009.

[2] H. J. Kushner and G. G. Yin. *Stochastic approximation and recursive algorithms and applications*. Springer-Verlag, second edition, 2003.

[3] O. Yu. Kul′chitskiĭ and A. È. Mozgovoĭ. An estimate for the rate of convergence of recurrent robust identification algorithms. *Kibernet. i Vychisl. Tekhn.*, 89:36–39, 1991.

[4] B. T. Polyak and A. B. Juditsky. Acceleration of stochastic approximation by averaging. *SIAM Journal on Control and Optimization*, 30(4):838–855, 1992.

[5] D. Ruppert. Efficient estimations from a slowly convergent Robbins-Monro process. Technical Report 781, Cornell University Operations Research and Industrial Engineering, 1988.

[6] V. Fabian. On asymptotic normality in stochastic approximation. *The Annals of Mathematical Statistics*, 39(4):1327–1332, 1968.

[7] Y. Nesterov and J. P. Vial. Confidence level solutions for stochastic programming. *Automatica*, 44(6):1559–1568, 2008.

[8] A. Nemirovski, A. Juditsky, G. Lan, and A. Shapiro. Robust stochastic approximation approach to stochastic programming. *SIAM Journal on Optimization*, 19(4):1574–1609, 2009.

[9] L. Bottou and Y. Le Cun. On-line learning for very large data sets. *Applied Stochastic Models in Business and Industry*, 21(2):137–151, 2005.

[10] L. Bottou and O. Bousquet. The tradeoffs of large scale learning. In *Advances in Neural Information Processing Systems (NIPS), 20*, 2008.

[11] S. Shalev-Shwartz and N. Srebro. SVM optimization: inverse dependence on training set size. In *Proc. ICML*, 2008.

[12] S. Shalev-Shwartz, Y. Singer, and N. Srebro. Pegasos: Primal estimated sub-gradient solver for svm. In *Proc. ICML*, 2007.

[13] S. Shalev-Shwartz, O. Shamir, N. Srebro, and K. Sridharan. Stochastic convex optimization. In *Conference on Learning Theory (COLT)*, 2009.

[14] L. Xiao. Dual averaging methods for regularized stochastic learning and online optimization. *Journal of Machine Learning Research*, 9:2543–2596, 2010.

[15] J. Duchi and Y. Singer. Efficient online and batch learning using forward backward splitting. *Journal of Machine Learning Research*, 10:2899–2934, 2009.

[16] J. M. Borwein and A. S. Lewis. *Convex Analysis and Nonlinear Optimization: Theory and Examples*. Springer, 2006.

[17] R. Durrett. *Probability: theory and examples*. Duxbury Press, third edition, 2004.

[18] B. Schölkopf and A. J. Smola. *Learning with Kernels*. MIT Press, 2001.

[19] J. Shawe-Taylor and N. Cristianini. *Kernel Methods for Pattern Analysis*. Cambridge University Press, 2004.

[20] Y. Nesterov. *Introductory lectures on convex optimization: a basic course*. Kluwer Academic Publishers, 2004.

[21] K. Sridharan, N. Srebro, and S. Shalev-Shwartz. Fast rates for regularized objectives. *Advances in Neural Information Processing Systems*, 22, 2008.

[22] N. N. Vakhania, V. I. Tarieladze, and S. A. Chobanyan. *Probability distributions on Banach spaces*. Reidel, 1987.

[23] F. Bach and E. Moulines. Non-asymptotic analysis of stochastic approximation algorithms for machine learning. Technical Report 00608041, HAL, 2011.

[24] A.S. Nemirovsky and D.B. Yudin. *Problem complexity and method efficiency in optimization*. Wiley & Sons, 1983.

[25] A. Agarwal, P. L. Bartlett, P. Ravikumar, and M. J. Wainwright. Information-theoretic lower bounds on the oracle complexity of convex optimization, 2010. Tech. report, Arxiv 1009.0571.

[26] E. Hazan and S. Kale. Beyond the regret minimization barrier: an optimal algorithm for stochastic strongly-convex optimization. In *Proc. COLT*, 2001.

[27] G. Casella and R. L. Berger. *Statistical Inference*. Duxbury Press, 2001.

[28] I. Steinwart. On the influence of the kernel on the consistency of support vector machines. *Journal of Machine Learning Research*, 2:67–93, 2002.

